# A Probabilistic Model for Generating Realistic Lip Movements from Speech

**Gwenn Englebienne**
School of Computer Science
University of Manchester
ge@cs.man.ac.uk

**Tim F. Cootes**
Imaging Science and Biomedical Engineering
University of Manchester
Tim.Cootes@manchester.ac.uk

**Magnus Rattray**
School of Computer Science
University of Manchester
magnus.rattray@manchester.ac.uk

## Abstract

The present work aims to model the correspondence between facial motion and speech. The face and sound are modelled separately, with phonemes being the link between both. We propose a sequential model and evaluate its suitability for the generation of the facial animation from a sequence of phonemes, which we obtain from speech. We evaluate the results both by computing the error between generated sequences and real video, as well as with a rigorous double-blind test with human subjects. Experiments show that our model compares favourably to other existing methods and that the sequences generated are comparable to real video sequences.

## 1 Introduction

Generative systems that model the relationship between face and speech offer a wide range of exciting prospects. Models combining speech and face information have been shown to improve automatic speech recognition [4]. Conversely, generating video-realistic animated faces from speech has immediate applications to the games and movie industries. There is a strong correlation between lip movements and speech [7,10], and there have been multiple attempts at generating an animated face to match some given speech realistically [2,3,9,13]. Studies have indicated that speech might be informative not only of lip movement but also of movement in the upper regions of the face [3]. Incorporating speech therefore seems crucial to the generation of true-to-life animated faces.

Our goal is to build a generative probabilistic model, capable of generating realistic facial animations in real time, given speech. We first use an Active Appearance Model (AAM [6]) to extract features from the video frames. The AAM itself is generative and allows us to produce video-realistic frames from the features. We then use a Hidden Markov Model (HMM [12]) to align phoneme labels to the audio stream of video sequences, and use this information to label the corresponding video frames. We propose a model which, when trained on these labelled video frames, is capable of generating new, realistic video from unseen phoneme sequences. Our model is a modification of Switching Linear Dynamical Systems (SLDS [1,15]) and we show that it performs better at generation than other existing models. We compare its performance to two previously proposed models by comparing the sequences they generate to a golden standard, features from real video sequences, and by asking volunteers to select the "real" video in a forced-choice test.

The results of human evaluation of our generated sequences are extremely encouraging. Our system performs well with any speech, and since it can easily handle real-time generation of the facial animation, it brings a realistic-looking, talking avatar within reach.

## 2 The Data

We used sequences from the freely available on-line news broadcast *Democracy Now!* The show is broadcast every weekday in a high quality MP4 format, and as such constitutes a constant source of new data. The text transcripts are available on-line, thus greatly facilitating the training of a speech recognition system. We manually extracted short video sequences of the news presenter talking (removing any inserts, telephone interviews, *etc.*), cutting at "natural" positions in the stream, *viz.* during pauses for breath and silences. The sequences are all of the same person, albeit on different days within a period of slightly more than a month. There was no reason to restrict the data to a single person, other than the difficulty to obtain sequences of similar quality from other sources.

All usable sequences were extracted from the data, that is, those where the face of the speaker was visible and the sound was not corrupted by external sound sources. The sequences do include hesitations, corrections, incomplete words, noticeable fatigue, breath, swallowing, *etc.* The speaker visibly makes an effort to speak clearly, but obviously makes no effort to reduce head motion or facial expression, and the data is hence probably as representative of the problem as can be hoped for.

In total, sequences totalling 1 hour and 7 minutes of video were extracted and annotated.[1] The data was split into independent training and test sets for a 10-fold cross validation, based on the number of sequences in each set (rather than the total amount of data). This resulted in training sets of an average of 60 minutes of data, and test sets of approximately 7 min. All models evaluated here were trained and tested on the same data sets.

**Sound features and labelling.** The sequences are split into an audio and a video stream, which are treated separately (see Figure 1). From the sound stream, we extract Mel Frequency Cepstrum Coefficients (MFCC) at a rate of 100Hz, using tools from the HMM Tool Kit [16], resulting in 13-dimensional feature vectors. We train a HMM on these MFCC features, and use it to align phonetic labels to the sound. This is an easier task than unrestricted speech recognition, and is done satisfactorily by a simple HMM with monophones as hidden states, where mixtures of Gaussian distributions model the emission densities. The sound samples are labelled with the Viterbi path through the HMM that was "unrolled" with the phonetic transcription of the text.

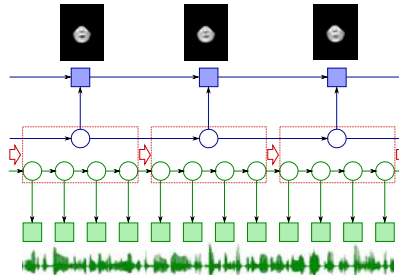

Figure 1: Combining sound and face

The labels obtained from the sound stream are then used to label the corresponding video frames. The difference in rate (the video is processed at 29.97 frames per second while MFCC coefficients are computed at 100 Hz) is handled by simple voting: each video frame is labelled with the phoneme that labels most of the corresponding sound frames.

**Face features.** The feature extraction for the video was done using an Active Appearance Model (AAM [6]). The AAM represents both the shape and the texture of an object in an image. The shape of the lower part of the face is represented by the location of 23 points on key features on the eyes, mouth and jaw-line (see Figure 2). Given the position of the points in a set of training images, we align them to a common co-ordinate frame and apply PCA to learn a low-dimensional linear model capturing the shape change [5]. The intensities across the region in each example are warped to the mean shape using a simple triangulation of the region (Fig 2), and PCA applied to the vectors of intensities sampled from each image. This leads to a low-dimensional linear model of the intensities in the mean frame. Efficient algorithms exist for matching such models to new images [6]. By combining shape and intensity model together, a wide range of convincing synthetic faces can be generated [6]. In this case a 32 parameter model proves sufficient. This is closely related to eigenfaces [14] but gives far better results as shape and texture are decoupled [8]. Since the AAM parameters

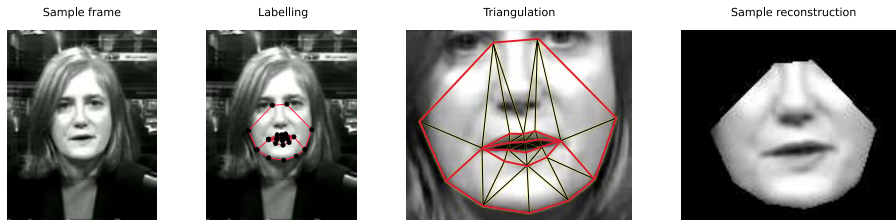

Figure 2: The face was modelled with an AAM. A set of training images is manually labelled as shown in the two leftmost images. A statistical model of the shape is then combined with a model of the texture within the triangles between feature points. Applying the model to a new image results in a vector of coefficients, which can be used to reconstruct the original image.

are a low-dimensional linear projection of the original object, projecting those parameters back to the high-dimensional space allows us to reconstruct the modelled part of the original image.

## 3    Modelling the dynamics of the face

We model the face using only phoneme labels to capture the shared information between speech and face. We use 41 distinct phoneme labels, two of which are reserved for breath and silence, the rest being the generally accepted phonemes in the English language. Most earlier techniques that use discrete labels to generate synthetic video sequences use some form of smooth interpolation between key frames [2, 9]. This requires finding the correct key frames, and lacks the flexibility of a probabilistic formulation. Brand uses a HMM where Gaussian distributions are fitted to a concatenation of the data features and "delta" features [3]. Since the distribution is fitted to both the features and the difference between features, the resulting "distribution" cannot be sampled, as it would result in non-sensical mismatch between features and delta features. It is therefore not genuinely generative and obtaining new sequences from the model requires solving an optimisation problem.

Under Brand's approach, new sequences are obtained by finding the most likely sequence of observations for a set of labels. This is done by setting the first derivative of the likelihood with respect to the observations to zero, resulting in a set of linear equations involving, at each time $t$, the observation $\mathbf{y}_t^s$ and the previous observation $\mathbf{y}_{t-1}^s$. Such a set of linear equations can be solved relatively efficiently thanks to its block-band-diagonal structure. This requires the storage of $\mathcal{O}(d^2 T)$ elements and $\mathcal{O}(d^3 T)$ time to solve, where $d$ is twice the dimensionality of the face features and $T$ is the number of frames in a sequence. This becomes non-trivial for sequences exceeding a few tens of seconds. More important, however, is that this cannot be done in real time, as the last label of the sequence must be known before the first observation can be computed.

In this work, we consider more standard probabilistic models of sequential data, which are genuinely generative. These models are shown to outperform Brand's approach for the generation of realistic sequences.

**Switching Linear Dynamical Systems.**  Before introducing the SLDS, we introduce some notational conventions. We have a set of $S$ video sequences, which we index with $s \in [1 \ldots S]$. The feature vector of the frame at time $t$ in the video sequence $s$ is indicated as $\mathbf{y}_t^s \in \mathbb{R}^d$, and the complete set of feature vectors for that sequence is denoted as $\{\mathbf{y}\}_1^{T_s}$, where $T_s$ is the length of the sequence. Continuous hidden variables are indicated as $\mathbf{x}$ and discrete state labels are indicated with $\pi$, where $\pi \in [1 \ldots \Pi]$.

In an SLDS, the sequence of observations $\{\mathbf{y}\}_1^{T_s}$ is modelled as being a noisy version of a hidden sequence $\{\mathbf{x}\}_1^{T_s}$ which depends on a sequence of discrete labels $\{\pi\}_1^{T_s}$. Each state $\pi$ is associated with a transition matrix $\mathbf{A}_\pi$ and with a distribution for the output noise $\mathbf{v}$ and the process noise $\mathbf{w}$, such that $\mathbf{y}_t^s = \mathbf{B}_{\pi_t^s}\mathbf{x}_t^s + \mathbf{v}_t$, $\mathbf{x}_1^s \sim \mathcal{N}(\boldsymbol{\mu}_{\pi_1^s}, \boldsymbol{\Sigma}_{\pi_1^s})$ and $\mathbf{x}_t^s = \mathbf{A}_{\pi_t^s}\mathbf{x}_{t-1}^s + \boldsymbol{\nu}_{\pi_t^s} + \mathbf{w}_t$ for $2 \leqslant t \leqslant T_s$. Both the output noise $\mathbf{v}_t$ and the process noise $\mathbf{w}_t$ are normally distributed with zero mean; $\mathbf{v}_t \sim \mathcal{N}(\mathbf{0}, \mathbf{R}_{\pi_t^s})$ and $\mathbf{w}_t \sim \mathcal{N}(\mathbf{0}, \mathbf{Q}_{\pi_t^s})$. The states in our application are

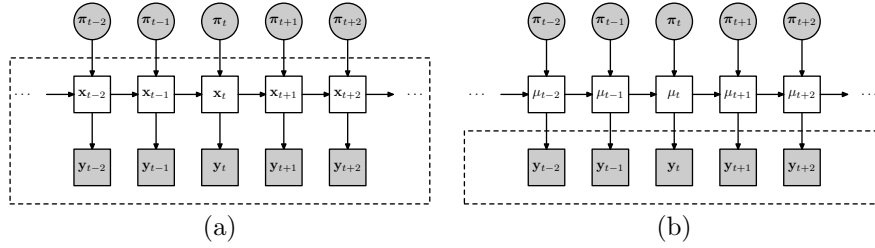

Figure 3: Graphical representation of the different models: figure (a) depicts the dependencies in an SLDS when the labels are known and (b) represents our proposed DPDS, where we assume the process is noiseless. The circles are discrete and the squares are multivariate continuous quantities. The shaded elements are observed and the random variables in the dashed box are conditioned on the quantities outside of it.

the phonemes, which are obtained from the sound. Notice that in general, when the state labels are not known, computing the likelihood in an SLDS is intractable as it requires the enumeration of all possible state sequences, which is exponential in $T$ [1]. In our case, however, the state label $\pi_t^s$ of each frame is known from the sound and the likelihood can be computed with the same algorithm as for a standard Linear Dynamical Systems (LDS), which is linear in $T$. Parameter optimisation can therefore be carried out efficiently with a standard EM algorithm. Also note that neither SLDS or LDS are commonly described with the explicit state bias $\boldsymbol{\nu}_{\pi_t^s}$, as this can easily be emulated by augmenting each latent vector $\mathbf{x}_t^s$ with a 1 and incorporating $\boldsymbol{\nu}_{\pi_t^s}$ into $\mathbf{A}_{\pi_t^s}$. However, doing so prevents us from using a diagonal matrix for $\mathbf{A}_{\pi_t^s}$, and experience has shown that the state mean is crucial to good prediction while the lack of sufficient data or, as is the case with our data, the à priori known approximate independence of the data dimensions may make the reduction of the complexity of $\mathbf{A}_{\pi_t^s}, \mathbf{Q}_{\pi_t^s}$ and $\mathbf{R}_{\pi_t^s}$ warranted.

In this form, the model is over-parametrised; it can be simplified without any loss of generality either by fixing $\mathbf{Q}_{\pi_t^s}$ to the identity matrix $\mathbf{I}$ or, if there is no reason to use a different dimensionality for $\mathbf{x}$ and $\mathbf{y}$, by setting $\mathbf{B}_{\pi_t^s} = \mathbf{I}$. We did the latter, as this makes the resulting $\{\mathbf{x}\}_1^T$ easier to interpret and compare across the different models we evaluate here.

We trained a SLDS by maximum likelihood and used the model to generate new sequences of face observations for given sequences of labels. This was done by computing the most likely sequence of observations for the given set of labels. An in-depth evaluation of the trained SLDS model, when used to generate new video sequences, is given in section 4. This evaluation shows that SLDS is overly flexible: it appears to explain the data well and results in a very high likelihood, but does a poor job at generating realistic new sequences.

**Deterministic Process Dynamical System.** We reduced the complexity of the model by simplifying its covariance structure. If we set the output noise $\mathbf{v}_t$ of the SLDS to zero, leaving only process noise, we obtain the autoregressive hidden Markov model [11]. This model has the advantage that it can be trained using an EM algorithm when the state labels are unknown, but we find that it performs very poorly at data generation. If we set the process noise $\mathbf{w}_t = \mathbf{0}$, however, then we obtain a more useful model. The complete hidden sequence $\{\mathbf{x}\}_1^T$ is then determined exactly by the labels $\{\pi\}_1^T$. The log-likelihood $p(\{\mathbf{y}\}|\{\pi\})$ is given by

$$\log p(\{\mathbf{y}\}|\{\mathbf{x}\}) = -\tfrac{1}{2}\sum_{s=1}^{S}\Big[\log|\boldsymbol{\Sigma}_{\pi_1^s}| + (\mathbf{y}_1^s - \mathbf{x}_1^s)^\top\boldsymbol{\Sigma}_{\pi_1^s}^{-1}(\mathbf{y}_1^s - \mathbf{x}_1^s)+$$

$$\sum_{t=2}^{T_s}\Big(\log|\mathbf{R}_{\pi_t^s}| + (\mathbf{y}_t^s - \mathbf{x}_t^s)^\top\mathbf{R}_{\pi_t^s}^{-1}(\mathbf{y}_t^s - \mathbf{x}_t^s)\Big) + dT_s\log 2\pi\Big] \quad (1)$$

where $\mathbf{x}_1^s = \boldsymbol{\mu}_{\pi_1^s}$ and $\mathbf{x}_t^s = \mathbf{A}_{\pi_t^s}\mathbf{x}_{t-1}^s + \boldsymbol{\nu}_{\pi_t^s}$ for $t > 1$. We will now refer to this model as the Deterministic Process Dynamical System (DPDS, see Figure 3). In our implementation we

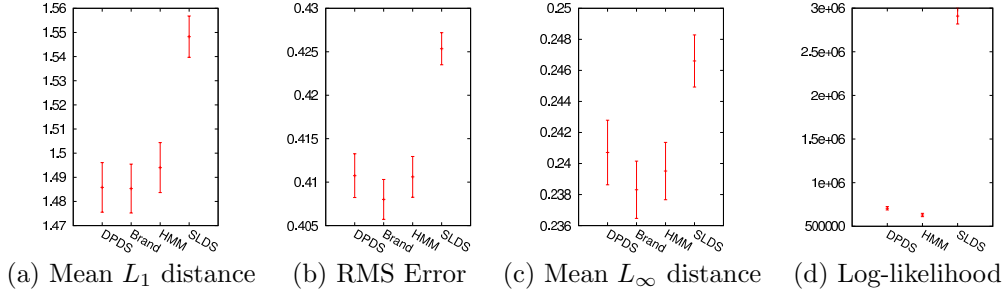

(a) Mean $L_1$ distance     (b) RMS Error     (c) Mean $L_\infty$ distance     (d) Log-likelihood

Figure 4: Comparison of the multiple models on the test data of 10-fold cross-validation. Each plot shows the mean error of the generated data with respect to the real data over the 10 folds. The error bars span the 95% confidence interval of the true error.

model all matrices $\mathbf{R}_{\pi_t^s}$, $\mathbf{\Sigma}_{\pi_t^s}$ as diagonal, and further reduce the complexity by sharing the output noise covariance over all states. It is reasonable to assume this because the features are the result of PCA and are therefore uncorrelated.

Since in this case the labels $\pi_t^s$ are known, equation (1) does not contain any hidden variables. Applying EM is therefore not necessary. Deriving a closed-form solution for the ML estimates of the parameters, however, results in solving polynomial equations of the order $T_s$, because $\mathbf{x}_t^s = f(\mathbf{A}_{\pi_2^s} \cdots \mathbf{A}_{\pi_t^s})$. An efficient solution is to use a gradient-based method. The log-likelihood of a sequence is a sum of scaled quadratic terms of $(\mathbf{y}_t^s - \mathbf{x}_t^s)$, where $\mathbf{x}_t^s = f(\{\pi\}_1^t)$. The log-likelihood must thus be computed by a forward iteration over all time steps $t$ using $\mathbf{x}_{t-1}^s$ to compute $\mathbf{x}_t^s$. The gradients of the likelihood with respect to $\mathbf{A}_{\pi_t^s}$ can be computed numerically in a similar fashion, by applying the chain rule iteratively at each time step and storing the result for the next step. The same could be done for other parameters, however for given values of $\mathbf{A}_{\pi_t^s}$, the values of $\boldsymbol{\mu}_{\pi_t^s}$, $\boldsymbol{\nu}_{\pi_t^s}$ and $\mathbf{R}_{\pi_t^s}$ that maximise the likelihood can be computed exactly by solving a set of linear equations. This markedly improves the rate of convergence. An algorithm for the computation of the gradients with respect to $\mathbf{A}_{\pi_t^s}$ and the exact evaluation of the other parameters is given in Appendix A.

**Sequence generation.** Since all models parametrise the distribution of the data, we can sample them to generate new observation sequences. In order to evaluate the performance of the models and compare it to Brand's model, it is however useful to generate the most likely sequence of observation features for a sequence of labels with the features of the corresponding real video sequence.

For both SLDS (when $\mathbf{B}_{\pi_t^s} = \mathbf{I}$) and the DPDS, the mean for a given sequence of labels $\{\pi\}_1^T$ is found by a forward iteration starting with $\hat{\mathbf{y}}_1 = \boldsymbol{\mu}_{\pi_1^s}$ and iterating for $t > 1$ with $\hat{\mathbf{y}}_t = \mathbf{A}_{\pi_t^s} \hat{\mathbf{y}}_{t-1} + \boldsymbol{\nu}_{\pi_t^s}$. This does not require the storage of the complete sequence in memory as the current observation only depends on the previous one. In setups where artificial speech is generated, the video sequence can therefore be generated at the same time as the audio sequence and without length limitations, with $\mathcal{O}(d)$ space and $\mathcal{O}(dT)$ time complexity, where $d$ is the dimensionality of the face features (without delta features).

## 4 Evaluation against real video

We evaluated the models in two ways: (1) by computing the error between generated face features and a ground truth (the features of real video), and (2) by asking human subjects to rate how they perceived the sequences. Both tests were done on the same real-world data, but partitioned differently: the comparison to the ground truth was done using 10-fold cross-validation, while the test on humans was done using a single partitioning, due to the limited availability of unbiased test subjects.

**Test error and likelihood.** In order to test the models against the ground truth, we use the sound to align the labels to the video and generate the corresponding face features. We use 10-fold cross validation and evaluate the performance of the models using different metrics, see Figure 4. Plot (a) shows, for different models, the $L_1$ error between the face

| A | prefer A | undecided | prefer B | B |
|---|---|---|---|---|
| Brand | 5 | 7 | 54 | **DPDS** |
| Brand | 4 | 7 | 55 | reality |
| Brand | 36 | 21 | 9 | SLDS |
| **DPDS** | 29 | 11 | 26 | reality |
| **DPDS** | 60 | 5 | 1 | SLDS |
| reality | 58 | 5 | 3 | SLDS |

$$\text{DPDS} \approx \text{reality} \succ \text{Brand} \succ \text{SLDS}$$

Table 1: Raw results of the Psychophysical test conducted by human volunteers. Every model is compared to every other model; the order in which models are listed in this table is meaningless. See text for details.

features generated for the test sound sequences and the face features extracted from the real video. We compared the sequences generated by DPDS, Brand's model and SLDS to the most likely observations under a standard HMM. This last model just generates the mean face for each phoneme, hence resulting in very unnatural sequences. It illustrates how an obviously incorrect model nevertheless performs very similarly to the other models in terms of generation error. Plots (b) and (c) respectively show the corresponding Root Mean Square (RMS) and $L_\infty$ error. We can see that, except for the SLDS which performs worse than the other methods in terms of $L_1$, RMS and $L_\infty$ error, the generation error for the models considered, under all metrics, is consistently not statistically significantly different.

In terms of the log-likelihood of the test data under the different models, the opposite is true: the traditional HMM and DPDS clearly perform worst, while SLDS performs dramatically better. The model with the highest likelihood generates the sequences with the largest error. The likelihood under Brand's model cannot be compared directly as it has double the amount of features. These results notwithstanding, great differences can be seen in the quality of the generated video sequences, and the models giving the lowest error or the highest likelihood are far from generating the most realistic sequences. We have therefore performed a rigorous test where volunteers were asked to evaluate the quality of the sequences.

**Psychophysical test.** For this experiment, we trained the models on a training set of 642 sequences of an average of 5 seconds each. We then labelled the sequences in our test set, which consists of 80 sequences and 436 seconds of video from sound with phonemes. These are substantial amounts of data, showing the face in a wide variety of positions.

We set up a web-based test, where 33 volunteers compared 12 pairs of video sequences. All video sequences had original sound, but the video stream was generated by any one of four methods: (1) from the face features extracted from the corresponding real video, (2) from SLDS, (3) from Brand's model and (4) from DPDS. A pool of 80 sequences was generated from previously unseen videos. The 12 pairs were chosen such that each generation method was pitted against each other generation method twice (once on each side, left or right, in order to eliminate bias towards a particular side) in random order. For each pair, corresponding sequences were chosen from the respective pools at random. The volunteers were only told that the sequences were either real or artificial, and were asked to either select the real video or to indicate that they could not decide. The test is kept available on-line for validation at `http://www.cs.manchester.ac.uk/ai/public/dpdseval`.

The results are shown in Table 1. The first row, *e.g.*, shows that when comparing Brand's model with the DPDS, people thought that the sequence generated with the former model was real in 5 cases, could not make up their mind in 7 cases, and thought the sequence generated with DPDS was real in 54 instances. These results indicate that DPDS performs quite well at generation, clearly much better than the two other models. Note however that this test discriminates the models very harshly. Despite the strong down-voting of Brand's model in this test, the sequences generated with that model do not look all that bad. They are over-smoothed, however, and humans appear to be very sensitive to that. Also remember that Brand's model is the only model considered here with a closed form solution for the parameter estimation given the labels. Contrary to the other two models, it can easily be trained in the absence of labelling, using an EM algorithm.

In order to correlate human judgement with the generation errors discussed at the start of this section, we have computed the same error measures on the data as partitioned for the psychophysical test. These confirmed the earlier conclusions: the SLDS, which humans like least, gives the highest likelihood and the worst generation errors while DPDS and Brand's model do not give significantly different errors.

## 5 Conclusion

In this work we have proposed a truly generative model, which allows real-time generation of talking faces given speech. We have evaluated it both using multiple error measures and with a thorough test of human perception. The latter test clearly shows that our method perceptually outperforms the others and is virtually indistinguishable from reality. Compared to Brand's method it is slower during training, and cannot easily be trained in the absence of labelling. This is a trade-off for the very fast generation and visually much more appealing face animation.

In addition, we have shown that traditional metrics do not agree with human perception. The error measures do not necessarily favour our method, but the human preference for it is very significant. We believe this deserves deeper analysis. In future work, we plan to investigate different error measures, especially on the more directly interpretable video frames rather than on the extracted features. We also intend to experiment with a covariance matrix per state and an unrestricted matrix structure for the transition matrix $\mathbf{A}_{\pi_t^s}$.

## Footnotes

[1]The data is publicly available at `http://www.cs.manchester.ac.uk/ai/public/demnow`.

## References

[1] David Barber. Expectation correction for smoothed inference in switching linear dynamical systems. *Journal of Machine Learning Research*, 7:2515–2540, 2006.

[2] V. Blanz, C. Basso, T. Poggio, and T. Vetter. Reanimating faces in images and video. In *Proceedings of ACM SIGGRAPH, Annual Conference Series*, 2003.

[3] M. Brand. Voice puppetry. In *SIGGRAPH '99: Proceedings of the 26th annual conference on Computer graphics and interactive techniques*, pages 21–28, New York, NY, USA, 1999. ACM Press/Addison-Wesley Publishing Co.

[4] C. Bregler, H. Hild, and S. Manke. Improving letter recognition by lipreading. In *Proceedings of ICASSP*, 1993.

[5] T. F. Cootes, C. J. Taylor, D. H. Cooper, and J. Graham. Active shape models, their training and application. *Comput. Vis. Image Underst.*, 61(1):38–59, 1995.

[6] T.F. Cootes, G.J. Edwards, and C.J. Taylor. Active appearance models. *IEEE Transactions on Pattern Analysis and Machine Intelligence*, 23(6):681–685, 2001.

[7] P. Duchnowski, U. Meier, and A. Weibel. See me, hear me: Integrating automatic speech recognition and lipreading. In *Proc. ICSLP 94*, 1994.

[8] G. Edwards, C. Taylor, and T. Cootes. Interpreting face images using active appearance models, 1998.

[9] T. F. Ezzat, G. Geiger, and T. Poggio. Trainable videorealistic speech animation. In *SIGGRAPH '02: Proceedings of the 29th annual conference on Computer graphics and interactive techniques*, pages 388–398, New York, NY, USA, 2002. ACM Press.

[10] H. McGurk and J. MacDonald. Hearing lips and seeing voices. *Nature*, pages 746 – 748, December 1976.

[11] Alan B. Poritz. Linear predictive hidden markov models and the speech signal. *Proc. IEEE Int. Conf. Acoustics, Speech and Signal Processing*, 7:1291–1294, May 1982.

[12] L. R. Rabiner. A tutorial on hidden markov models and selected applications in speech recognition. In *Readings in speech recognition*, pages 267–296. Morgan Kaufmann Publishers Inc., San Francisco, CA, USA, 1990.

[13] B. Theobald, G. Cawley, I. Matthews, J. Glauert, and J. Bangham. 2.5D visual speech synthesis using appearance models. *Proceedings of the British Machine Vision Conference*, 2003.

[14] M. A. Turk and A. P. Pentland. Face recognition using eigenfaces. *Proc. IEEE Conf. Computer Vision and Pattern Recognition*, pages 586–591, 1991.

[15] Mike West and Jeff Harrison. *Bayesian Forecasting and Dynamic Models.* Springer, 1999.

[16] S. Young. The HTK hidden markov model toolkit: Design and philosophy, 1993.

# A   Parameter estimation in DPDS

The log-likelihood of a sequence is given by eq. 1, which is a multiplicative function of $\mathbf{A}$ ($x_1 = f(\mathbf{A}_{\pi_1^s})$, $x_2 = f(\mathbf{A}_{\pi_2^s}\mathbf{A}_{\pi_1^s})$, *etc.*). Applying the chain rule repeatedly gives us, for diagonal matrices and using $\mathcal{L}_t$ to denote the log-likelihood of a single observation at time $t$, that $\partial \mathcal{L}_1 / \partial \mathbf{A}_n = 0$ and $\partial \mathcal{L}_t / \partial \mathbf{A}_n = \mathbf{R}_{\pi_t^s}^{-1}(\mathbf{y}_t^s - \mathbf{x}_t^s)(\partial \mathbf{x}_t^s / \partial \mathbf{A}_n)$ for $2 \leqslant t \leqslant T$, where

$$\frac{\partial \mathbf{x}_t^s}{\partial \mathbf{A}_n} = \mathbf{x}_t^s \delta_{n\pi_t^s} + \mathbf{A}_{\pi_t^s}\frac{\partial \mathbf{x}_{t-1}^s}{\partial \mathbf{A}_n}, \text{ and } \delta_{n\pi_t^s} = 1 \text{ iff } n = \pi_t^s \tag{2}$$

There we give the gradients for diagonal matrices for simplicity of notation and because we used diagonal matrices for this work, but the same principle applies to full matrices. The gradient of the likelihood is then $\partial \mathcal{L} / \partial \mathbf{A}_n = \sum_{s=1}^{S}\sum_{t=2}^{T_s} \partial \mathcal{L}_{s,t} / \partial \mathbf{A}_n$. In general the same is done for the other parameters of the model, however when the covariance is shared by all states, the value of the other parameters can be maximised exactly as described below. In the following, superscripts differentiate between variables by indicating what the variable is a coefficient to. The covariance $\mathbf{R} = \sum_{s=1}^{S}\sum_{t=2}^{T_s}(\mathbf{y}_t^s - \mathbf{x}_t^s)(\mathbf{y}_t^s - \mathbf{x}_t^s)^\top / \sum_{s=1}^{S}(T_s - 1)$ where $\mathbf{x}_1^s = \boldsymbol{\mu}_{\pi_1^s}$, $\mathbf{x}_t^s = \mathbf{A}_{\pi_t^s}\mathbf{x}_{t-1}^s + \boldsymbol{\nu}_{\pi_t^s}$, while $\boldsymbol{\mu}_{\pi_t^s}$ and $\boldsymbol{\nu}_{\pi_t^s}$ are found by solving the system of linear equations (3) for which the coefficients $\mathbf{D}$ and $\mathbf{b}$ are computed by Algorithm 1, which takes $\{\pi\}$, $\{\mathbf{y}\}$ and the current values of $\mathbf{A}_{1\ldots\Pi}$ as input:

$$\begin{bmatrix} \mathrm{diag}_{\Pi \times \Pi}(\mathbf{D}_n^{\mu\mu}) & \mathbf{D}_{\Pi \times \Pi}^{\mu\nu} \\ \hline \mathbf{D}_{\Pi \times \Pi}^{\nu\mu} & \mathbf{D}_{\Pi \times \Pi}^{\nu\nu} \end{bmatrix} \begin{bmatrix} \boldsymbol{\mu}_{\Pi \times 1} \\ \hline \boldsymbol{\nu}_{\Pi \times 1} \end{bmatrix} = \begin{bmatrix} \mathbf{b}_{\Pi \times 1}^{\mu} \\ \hline \mathbf{b}_{\Pi \times 1}^{\nu} \end{bmatrix} \text{ where } \mathbf{X}_{\Pi \times \Pi} \triangleq \begin{bmatrix} \mathbf{X}_{1,1} & \cdots & \mathbf{X}_{1,\Pi} \\ \hline \vdots & \ddots & \vdots \\ \hline \mathbf{X}_{1,\Pi} & \cdots & \mathbf{X}_{\Pi,\Pi} \end{bmatrix} \tag{3}$$

---

**Algorithm 1** Maximisation of $\mathcal{L}$ with respect to $\boldsymbol{\mu}$ and $\boldsymbol{\nu}$

---
  **for** $n \in \{1 \ldots \Pi\}$ **do**
    $\mathbf{b}_n^{\mu} \leftarrow \mathbf{0}$, $\mathbf{b}_n^{\nu} \leftarrow \mathbf{0}$, $\mathbf{D}_n^{\mu\mu} \leftarrow \mathbf{0}$
    $\forall m \in \{1 \ldots \Pi\}$: $\mathbf{D}_{n,m}^{\mu\nu}, \mathbf{D}_{n,m}^{\nu\nu}, \mathbf{D}_n^{\nu\mu} \leftarrow \mathbf{0}$
    **for** $s \in \{s | \pi_0^s = n\}$ **do**                  ▷ Compute coefficients $\mathbf{D}_n^{\mu\mu}, \mathbf{D}_{nx}^{\mu\nu}, \mathbf{b}_n^{\mu}$ to $\boldsymbol{\mu}_n$
        $\mathbf{D}_n^{\mu\mu} \leftarrow \mathbf{D}_n^{\mu\mu} + \mathbf{I}$, $\mathbf{D}^{\mu} \leftarrow \mathbf{I}$, $\mathbf{b}_n^{\mu} \leftarrow \mathbf{b}_n^{\mu} + \mathbf{y}_t^s$
        $\forall m \in \{1 \ldots \Pi\}$: $\mathbf{C}_m^{\mu} \leftarrow \mathbf{0}$         ▷ $\mathbf{C}_m^{\mu}$ and $\mathbf{D}^{\mu}$ below are temporary variables
        **for** $t \in \{2 \ldots T_s\}$ **do**
            $\mathbf{D}^{\mu} \leftarrow \mathbf{D}^{\mu} + \mathbf{A}_{\pi_t^s}\mathbf{D}^{\mu}$, $\mathbf{D}_n^{\mu\mu} \leftarrow \mathbf{D}_n^{\mu\mu} + \mathbf{D}^{\mu}\mathbf{D}^{\mu}$, $\mathbf{b}_n^{\mu} \leftarrow \mathbf{b}_n^{\mu} + \mathbf{D}^{\mu}\mathbf{y}_t^s$
            $\forall m \in \{1 \ldots \Pi\}$: $\mathbf{C}_m^{\mu} \leftarrow \mathbf{A}_{\pi_t^s}\mathbf{C}_m^{\mu}$, $\mathbf{D}_{n,m}^{\mu\nu} \leftarrow \mathbf{D}_{n,m}^{\mu\nu} + \mathbf{D}^{\mu}\mathbf{C}_m^{\mu}$
            $\mathbf{C}_{\pi_t^s}^{\mu} \leftarrow \mathbf{C}_{\pi_t^s}^{\mu} + \mathbf{I}$
        **end for**
    **end for**
    **for** $s \in \{1 \ldots S\}$ **do**                  ▷ Compute coefficients $\mathbf{D}_{nx}^{\nu\mu}, \mathbf{D}_{nx}^{\nu\nu}, \mathbf{b}_n^{\nu}$ to $\boldsymbol{\nu}_n$
        $\forall m \in \{1 \ldots \Pi\}$: $\mathbf{C}_m^{\nu} \leftarrow \mathbf{0}$
        $\mathbf{D}^{\nu} \leftarrow 0$, $\mathbf{C}^{\mu} \leftarrow \mathbf{I}$             ▷ $\mathbf{C}_m^{\nu}$, $\mathbf{D}^{\nu}$, $\mathbf{C}_m^{\mu}$ are temporary variables
        **for** $t \in \{2 \ldots T_s\}$ **do**
            $\mathbf{D}^{\nu} \leftarrow \mathbf{A}_{\pi_t^s}\mathbf{D}^{\nu}$,
            **if** $\pi_t^s = n$ **then**
                $\mathbf{D}^{\nu} \leftarrow \mathbf{D}^{\nu} + \mathbf{I}$
            **end if**
            $\forall m \in \{1 \ldots \Pi\}$: $\mathbf{C}_m^{\nu} \leftarrow \mathbf{A}_{\pi_t^s}\mathbf{C}_m^{\nu}$, $\mathbf{D}_{n,m}^{\nu\nu} \leftarrow \mathbf{D}_{n,m}^{\nu\nu} + \mathbf{D}^{\nu}\mathbf{C}_m^{\nu}$
            $\mathbf{C}_{\pi_t^s}^{\nu} \leftarrow \mathbf{C}_{\pi_t^s}^{\nu} + \mathbf{I}$, $\mathbf{C}^{\mu} \leftarrow \mathbf{A}_{\pi_t^s}\mathbf{C}^{\mu}$, $\mathbf{D}_{\pi_1^s}^{\nu\mu} \leftarrow \mathbf{D}_{\pi_1^s}^{\nu\mu} + \mathbf{D}^{\nu}\mathbf{C}^{\mu}$, $\mathbf{b}_n^{\nu} \leftarrow \mathbf{b}_n^{\nu} + \mathbf{D}^{\nu}\mathbf{y}_t^s$
        **end for**
    **end for**
  **end for**

---

